# Bayesian Image Super-Resolution

**Michael E. Tipping and Christopher M. Bishop**

Microsoft Research
Cambridge, CB3 0FB, U.K.
*{mtipping,cmbishop}@microsoft.com*
http://research.microsoft.com/~{mtipping,cmbishop}

## Abstract

The extraction of a single high-quality image from a set of low-resolution images is an important problem which arises in fields such as remote sensing, surveillance, medical imaging and the extraction of still images from video. Typical approaches are based on the use of cross-correlation to register the images followed by the inversion of the transformation from the unknown high resolution image to the observed low resolution images, using regularization to resolve the ill-posed nature of the inversion process. In this paper we develop a Bayesian treatment of the super-resolution problem in which the likelihood function for the image registration parameters is based on a marginalization over the unknown high-resolution image. This approach allows us to estimate the unknown point spread function, and is rendered tractable through the introduction of a Gaussian process prior over images. Results indicate a significant improvement over techniques based on MAP (maximum a-posteriori) point optimization of the high resolution image and associated registration parameters.

## 1 Introduction

The task in super-resolution is to combine a set of low resolution images of the same scene in order to obtain a single image of higher resolution. Provided the individual low resolution images have sub-pixel displacements relative to each other, it is possible to extract high frequency details of the scene well beyond the Nyquist limit of the individual source images.

Ideally the low resolution images would differ only through small (sub-pixel) translations, and would be otherwise identical. In practice, the transformations may be more substantial and involve rotations or more complex geometric distortions. In addition the scene itself may change, for instance if the source images are successive frames in a video sequence. Here we focus attention on static scenes in which the transformations relating the source images correspond to translations and rotations, such as can be obtained by taking several images in succession using a hand held digital camera. Our approach is readily extended to more general projective transformations if desired. Larger changes in camera position or orientation can be

handled using techniques of robust feature matching, constrained by the epipolar geometry, but such sophistication is unnecessary in the present context.

Most previous approaches, for example [1, 2, 3], perform an initial registration of the low resolution images with respect to each other, and then keep this registration fixed. They then formulate probabilistic models of the image generation process, and use maximum likelihood to determine the pixel intensities in the high resolution image. A more convincing approach [4] is to determine simultaneously both the low resolution image registration parameters and the pixel values of the high resolution image, again through maximum likelihood.

An obvious difficulty of these techniques is that if the high resolution image has too few pixels then not all of the available high frequency information is extracted from the observed images, whereas if it has too many pixels the maximum likelihood solution becomes ill conditioned. This is typically resolved by the introduction of penalty terms to regularize the maximum likelihood solution, where the regularization coefficients may be set by cross-validation. The regularization terms are often motivated in terms of a prior distribution over the high resolution image, in which case the solution can be interpreted as a MAP (maximum a-posteriori) optimization.

Baker and Kanade [5] have tried to improve the performance of super-resolution algorithms by developing domain-specific image priors, applicable to faces or text for example, which are learned from data. In this case the algorithm is effectively hallucinating perceptually plausible high frequency features. Here we focus on general purpose algorithms applicable to any natural image, for which the prior encodes only high level information such as the correlation of nearby pixels.

The key development in this paper, which distinguishes it from previous approaches, is the use of Bayesian, rather than simply MAP, techniques by *marginalizing* over the unknown high resolution image in order to determine the low resolution image registration parameters. Our formulation also allows the choice of continuous values for the up-sampling process, as well the shift and rotation parameters governing the image registration.

The generative process by which the high resolution image is smoothed to obtain a low resolution image is described by a point spread function (PSF). It has often been assumed that the point spread function is known in advance, which is unrealistic. Some authors [3] have estimated the PSF in advance using only the low resolution image data, and then kept this estimate fixed while extracting the high resolution image. A key advantage of our Bayesian marginalization is that it allows us to determine the point spread function alongside both the registration parameters and the high resolution image in a single, coherent inference framework.

As we show later, if we attempt to determine the PSF as well as the registration parameters and the high resolution image by joint optimization, we obtain highly biased (over-fitted) results. By marginalizing over the unknown high resolution image we are able to determine the PSF and the registration parameters accurately, and thereby reconstruct the high resolution image with subjectively very good quality.

## 2 Bayesian Super-resolution

Suppose we are given $K$ low-resolution intensity images (the extension to 3-colour images is straightforward). We shall find it convenient notationally to represent the images as vectors $\mathbf{y}^{(k)}$ of length $M$, where $k = 1, \ldots, K$, obtained by raster scanning the pixels of the images. Each image is shifted and rotated relative to a

reference image which we shall arbitrarily take to be $\mathbf{y}^{(1)}$. The shifts are described by 2-dimensional vectors $\mathbf{s}_k$, and the rotations are described by angles $\theta_k$.

The goal is to infer the underlying scene from which the low resolution images are generated. We represent this scene by a single high-resolution image, which we again denote by a raster-scan vector $\mathbf{x}$ whose length is $N \gg M$.

Our approach is based on a generative model for the observed low resolution images, comprising a prior over the high resolution image together with an observation model describing the process by which a low resolution image is obtained from the high resolution one.

It should be emphasized that the real scene which we are trying to infer has effectively an infinite resolution, and that its description as a pixellated image is a computational artefact. In particular if we take the number $N$ of pixels in this image to be large the inference algorithm should remain well behaved. This is not the case with maximum likelihood approaches in which the value of $N$ must be limited to avoid ill-conditioning. In our approach, if $N$ is large the correlation of neighbouring pixels is determined primarily by the prior, and the value of $N$ is limited only by the computational cost of working with large numbers of high resolution pixels.

We represent the prior over the high resolution image by a Gaussian process

$$p(\mathbf{x}) = \mathcal{N}(\mathbf{x}|\mathbf{0}, \mathbf{Z}_x) \tag{1}$$

where the covariance matrix $\mathbf{Z}_x$ is chosen to be of the form

$$Z_x(i, j) = A \exp\left\{-\frac{\|\mathbf{v}_i - \mathbf{v}_j\|^2}{r^2}\right\}. \tag{2}$$

Here $\mathbf{v}_i$ denotes the spatial position in the 2-dimensional image space of pixel $i$, the coefficient $A$ measures the 'strength' of the prior, and $r$ defines the correlation length scale. Since we take $\mathbf{Z}_x$ to be a fixed matrix, it is straightforward to use a different functional form for $\mathbf{Z}_x$ if desired. It should be noted that in our image representation the pixel intensity values lie in the range $(-0.5, 0.5)$, and so in principle a Gaussian process prior is inappropriate[1]. In practice we have found that this causes little difficulty, and in Section 4 we discuss how a more appropriate distribution could be used.

The low resolution images are assumed to be generated from the high resolution image by first applying a shift and a rotation, then convolving with some point spread function, and finally downsampling to the lower resolution. This is expressed through the transformation equation

$$\mathbf{y}^{(k)} = \mathbf{W}^{(k)}\mathbf{x} + \boldsymbol{\epsilon}^{(k)} \tag{3}$$

where $\boldsymbol{\epsilon}^{(k)}$ is a vector of independent Gaussian random variables $\epsilon_i \sim \mathcal{N}(0, \beta^{-1})$, with zero mean and precision (inverse variance) $\beta$, representing noise terms intended to model the camera noise as well as to capture any discrepancy between our generative model and the observed data.

The transformation matrix $\mathbf{W}^{(k)}$ in (3) is given by a point spread function which captures the down-sampling process and which we again take to have a 'Gaussian' form

$$W_{ji}^{(k)} = \widetilde{W}_{ji}^{(k)} / \sum_{i'} \widetilde{W}_{ji'}^{(k)} \tag{4}$$

with

$$\widetilde{W}_{ji}^{(k)} = \exp\left\{-\frac{\|\mathbf{v}_i - \mathbf{u}_j^{(k)}\|^2}{\gamma^2}\right\} \tag{5}$$

where $j = 1, \ldots M$ and $i = 1, \ldots, N$. Here $\gamma$ represents the 'width' of the point spread function, and we shall treat $\gamma$ as an unknown parameter to be determined from the data. Note that our approach generalizes readily to any other form of point spread function, possibly containing several unknown parameters, provided it is differentiable with respect to those parameters.

In (5) the vector $\mathbf{u}_j^{(k)}$ is the centre of the PSF and is dependent on the shift and rotation of the low resolution image. We choose a parameterization in which the centre of rotation coincides with the centre $\overline{\mathbf{v}}$ of the image, so that

$$\mathbf{u}_j^{(k)} = \mathbf{R}^{(k)}(\mathbf{v}_j - \overline{\mathbf{v}}) + \overline{\mathbf{v}} + \mathbf{s}_k \tag{6}$$

where $\mathbf{R}^{(k)}$ is the rotation matrix

$$\mathbf{R}^{(k)} = \begin{pmatrix} \cos\theta_k & \sin\theta_k \\ -\sin\theta_k & \cos\theta_k \end{pmatrix}. \tag{7}$$

We can now write down the likelihood function in the form

$$p(\mathbf{y}^{(k)}|\mathbf{x}, \mathbf{s}_k, \theta_k, \gamma) = \left(\frac{\beta}{2\pi}\right)^{M/2} \exp\left\{-\frac{\beta}{2}\|\mathbf{y}^{(k)} - \mathbf{W}^{(k)}\mathbf{x}\|^2\right\}. \tag{8}$$

Assuming the images are generated independently from the model, we can then write the posterior distribution over the high resolution image in the form

$$p(\mathbf{x}|\{\mathbf{y}^{(k)}, \mathbf{s}_k, \theta_k\}, \gamma) = \frac{p(\mathbf{x})\prod_{k=1}^{K} p(\mathbf{y}^{(k)}|\mathbf{x}, \mathbf{s}_k, \theta_k, \gamma)}{p(\{\mathbf{y}^{(k)}\}|\{\mathbf{s}_k, \theta_k\}, \gamma)}, \tag{9}$$

$$= \mathcal{N}(\boldsymbol{\mu}, \boldsymbol{\Sigma}), \tag{10}$$

with

$$\boldsymbol{\Sigma} = \left[\mathbf{Z}_x^{-1} + \beta\left(\sum_{k=1}^{K} \mathbf{W}^{(k)\mathrm{T}}\mathbf{W}^{(k)}\right)\right]^{-1}, \tag{11}$$

$$\boldsymbol{\mu} = \beta\boldsymbol{\Sigma}\left(\sum_{k=1}^{K} \mathbf{W}^{(k)\mathrm{T}}\mathbf{y}^{(k)}\right). \tag{12}$$

Thus the posterior distribution over the high resolution image is again a Gaussian process.

If we knew the registration parameters $\{\mathbf{s}_k, \theta_k\}$, as well as the PSF width parameter $\gamma$, then we could simply take the mean $\boldsymbol{\mu}$ (which is also the maximum) of the posterior distribution to be our super-resolved image. However, the registration parameters are unknown. Previous approaches have either performed a preliminary registration of the low resolution images against each other and then fixed the registration while determining the high resolution image, or else have maximized the posterior distribution (9) jointly with respect to the high resolution image $\mathbf{x}$ and the registration parameters (which we refer to as the 'MAP' approach). Neither approach takes account of the uncertainty in determining the high resolution image and the consequential effects on the optimization of the registration parameters.

Here we adopt a Bayesian approach by marginalizing out the unknown high resolution image. This gives the marginal likelihood function for the low resolution images in the form

$$p(\mathbf{y}|\{\mathbf{s}_k, \theta_k\}, \gamma) = \mathcal{N}(\mathbf{0}, \mathbf{Z}_y) \tag{13}$$

where

$$\mathbf{Z}_y = \beta^{-1}\mathbf{I} + \mathbf{W}\mathbf{Z}_x\mathbf{W}^\mathsf{T}, \tag{14}$$

and $\mathbf{y}$ and $\mathbf{W}$ are the vector and matrix of stacked $\mathbf{y}^{(k)}$ and $\mathbf{W}^{(k)}$ respectively. Using some standard matrix manipulations we can rewrite the marginal likelihood in the form

$$\log p(\mathbf{y}|\{\mathbf{s}_k, \theta_k\}, \gamma) = -\frac{1}{2}\left[\beta\sum_{k=1}^{K}\|\mathbf{y}^{(k)} - \mathbf{W}^{(k)}\boldsymbol{\mu}\|^2 + \boldsymbol{\mu}^\mathsf{T}\mathbf{Z}_x^{-1}\boldsymbol{\mu}\right.$$
$$\left. + \log|\mathbf{Z}_x| - \log|\boldsymbol{\Sigma}| - KM\log\beta\right]. \tag{15}$$

We now wish to optimize this marginal likelihood with respect to the parameters $\{\mathbf{s}_k, \theta_k\}, \gamma$, and to do this we have compared two approaches. The first is to use the expectation-maximization (EM) algorithm. In the E-step we evaluate the posterior distribution over the high resolution image given by (10). In the M-step we maximize the expectation over $\mathbf{x}$ of the log of the complete data likelihood $p(\mathbf{y}, \mathbf{x}|\{\mathbf{s}_k, \theta_k\}, \gamma)$ obtained from the product of the prior (1) and the likelihood (8). This maximization is done using the scaled conjugate gradients algorithm (SCG) [6]. The second approach is to maximize the marginal likelihood (15) directly using SCG. Empirically we find that direct optimization is faster than EM, and so has been used to obtain the results reported in this paper.

Since in (15) we must compute $\boldsymbol{\Sigma}$, which is $N \times N$, in practice we optimize the shift, rotation and PSF width parameters based on an appropriately-sized subset of the image only. The complete high resolution image is then found as the mode of the full posterior distribution, obtained iteratively by maximizing the numerator in (9), again using SCG optimization.

## 3 Results

In order to evaluate our approach we first apply it to a set of low resolution images synthetically down-sampled (by a linear scaling of 4 to 1, or 16 pixels to 1) from a known high-resolution image as follows. For each image we wish to generate we first apply a shift drawn from a uniform distribution over the interval $(-2, 2)$ in units of high resolution pixels (larger shifts could in principle be reduced to this level by pre-registering the low resolution images against each other) and then apply a rotation drawn uniformly over the interval $(-4, 4)$ in units of degrees. Finally we determine the value at each pixel of the low resolution image by convolution of the original image with the point spread function (centred on the low resolution pixel), with width parameter $\gamma = 2.0$. From a high-resolution image of $384 \times 256$ we chose to use a set of 16 images of resolution $96 \times 64$.

In order to limit the computational cost we use patches from the centre of the low resolution image of size $9 \times 9$ in order to determine the values of the shift, rotation and PSF width parameters. We set the resolution of the super-resolved image to have 16 times as many pixels as the low resolution images which, allowing for shifts and the support of the point spread function, gives $N = 50 \times 50$. The Gaussian process prior is chosen to have width parameter $r = 1.0$, variance parameter $A = $

0.04, and the noise process is given a standard deviation of 0.05. Note that these values can be set sensibly *a priori* and need not be tuned to the data.

The scaled conjugate gradient optimization is initialised by setting the shift and rotation parameters equal to zero, while the PSF width $\gamma$ is initialized to 4.0 since this is the upsampling factor we have chosen between low resolution and super-resolved images. We first optimize only the shifts, then we optimize both shifts and rotations, and finally we optimize shifts, rotations and PSF width, in each case running until a suitable convergence tolerance is reached.

In Figure 1(a) we show the original image, together with an example low resolution image in Figure 1(b). Figure 1(c) shows the super-resolved image obtained using our Bayesian approach. We see that the super-resolved image is of dramatically better quality than the low resolution images from which it is inferred. The converged value for the PSF width parameter is $\gamma = 1.94$, close to the true value 2.0.

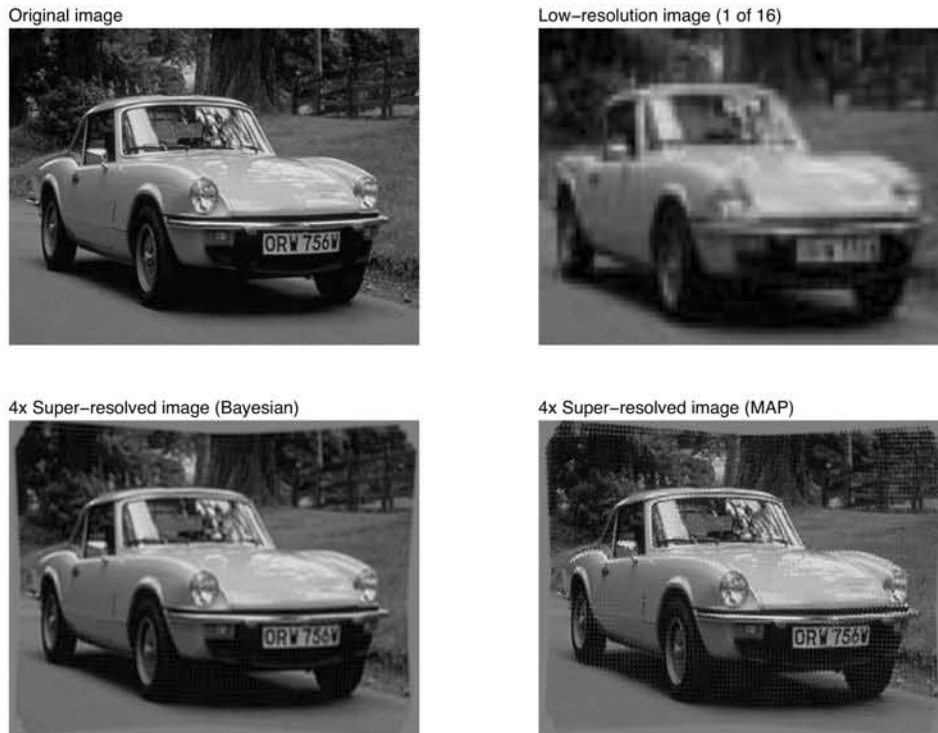

**Figure 1:** Example using synthetically generated data showing (*top left*) the original image, (*top right*) an example low resolution image and (*bottom left*) the inferred super-resolved image. Also shown, in (*bottom right*), is a comparison super-resolved image obtained by joint optimization with respect to the super-resolved image and the parameters, demonstrating the significantly poorer result.

Notice that there are some small edge effects in the super-resolved image arising from the fact that these pixels only receive evidence from a subset of the low resolution images due to the image shifts. Thus pixels near the edge of the high resolution image are determined primarily by the prior.

For comparison we show, in Figure 1(d), the corresponding super-resolved image obtained by performing a MAP optimization with respect to the high resolution image. This is of significantly poorer quality than that obtained from our Bayesian approach. The converged value for the PSF width in this case is $\gamma = 0.43$ indicating severe over-fitting.

In Figure 2 we show plots of the true and estimated values for the shift and rotation parameters using our Bayesian approach and also using MAP optimization. Again we see the severe over-fitting resulting from joint optimization, and the significantly better results obtained from the Bayesian approach.

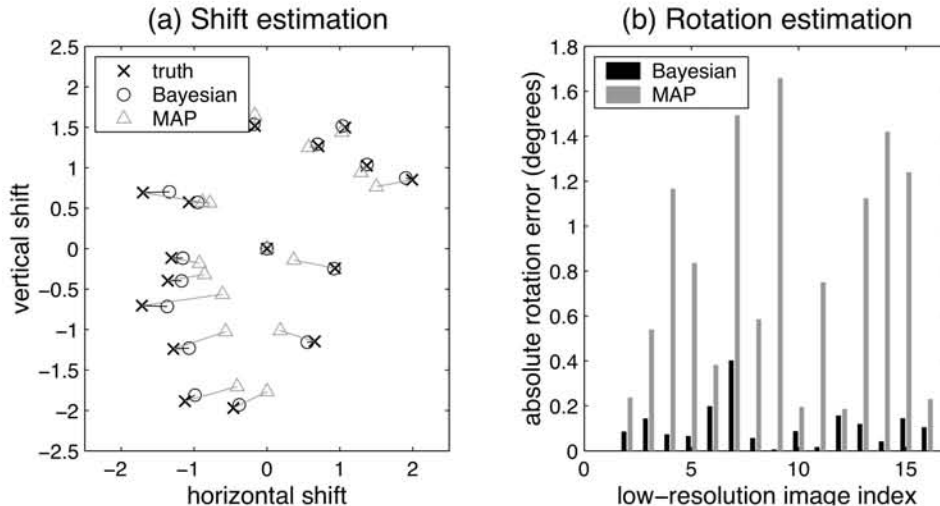

**Figure 2:** (a) Plots of the true shifts for the synthetic data, together with the estimated values obtained by optimization of the marginal likelihood in our Bayesian framework and for comparison the corresponding estimates obtained by joint optimization with respect to registration parameters and the high resolution image. (b) Comparison of the errors in determining the rotation parameters for both Bayesian and MAP approaches.

Finally, we apply our technique to a set of images obtained by taking 16 frames using a hand held digital camera in 'multi-shot' mode (press and hold the shutter release) which takes about 12 seconds. An example image, together with the super-resolved image obtained using our Bayesian algorithm, is shown in Figure 3.

## 4 Discussion

In this paper we have proposed a new approach to the problem of image super-resolution, based on a marginalization over the unknown high resolution image using a Gaussian process prior. Our results demonstrate a worthwhile improvement over previous approaches based on MAP estimation, including the ability to estimate parameters of the point spread function.

One potential application our technique is the extraction of high resolution images from video sequences. In this case it will be necessary to take account of motion blur, as well as the registration, for example by tracking moving objects through the successive frames [7].

(a) Low-resolution image (1 of 16)    (b) 4x Super-resolved image (Bayesian)

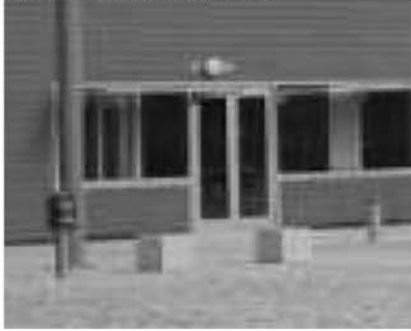 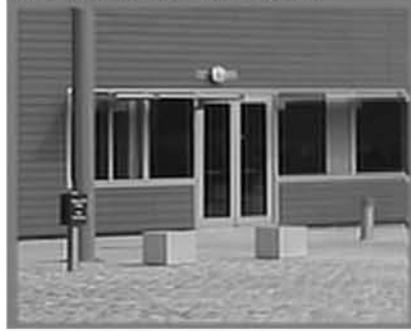

**Figure 3:** Application to real data showing in (a) one of the 16 captured in succession usind a hand held camera of a doorway with nearby printed sign. Image (b) shows the final image obtained from our Bayesian super-resolution algorithm.

Finally, having seen the advantages of marginalizing with respect to the high resolution image, we can extend this approach to a fully Bayesian one based on Markov chain Monte Carlo sampling over all unknown parameters in the model. Since our model is differentiable with respect to these parameters, this can be done efficiently using the hybrid Monte Carlo algorithm. This approach would allow the use of a prior distribution over high resolution pixel intensities which was confined to a bounded interval, instead of the Gaussian assumed in this paper. Whether the additional improvements in performance will justify the extra computational complexity remains to be seen.

## Footnotes

[1] Note that the established work we have referenced, where a Gaussian prior or quadratic regularlizer is utilised, also overlooks the bounded nature of the pixel space.

# References

[1] N. Nguyen, P. Milanfar, and G. Golub. A computationally efficient superresolution image reconstruction algorithm. *IEEE Transactions on Image Processing*, 10(4):573–583, 2001.

[2] V. N. Smelyanskiy, P. Cheeseman, D. Maluf, and R. Morris. Bayesian super-resolved surface reconstruction from images. In *Proceedings CVPR*, volume 1, pages 375–382, 2000.

[3] D. P. Capel and A. Zisserman. Super-resolution enhancement of text image sequences. In *International Conference on Pattern Recognition*, pages 600–605, Barcelona, 2000.

[4] R. C. Hardie, K. J. Barnard, and E. A. Armstrong. Joint MAP registration and high-resolution image estimation using a sequence of undersampled images. *IEEE Transactions on Image Processing*, 6(12):1621–1633, 1997.

[5] S. Baker and T. Kanade. Limits on super-resolution and how to break them. Technical report, Carnegie Mellon University, 2002. submitted to IEEE Transactions on Pattern Analysis and Machine Intelligence.

[6] I. T. Nabney. *Netlab: Algorithms for Pattern Recognition*. Springer, London, 2002. http://www.ncrg.aston.ac.uk/netlab/.

[7] B. Bascle, A. Blake, and A. Zisserman. Motion deblurring and super-resolution from an image sequence. In *Proceedings of the Fourth European Conference on Computer Vision*, pages 573–581, Cambridge, England, 1996.
